# Learning Spatio-Temporal Planning from a Dynamic Programming Teacher: Feed-Forward Neurocontrol for Moving Obstacle Avoidance

**Gerald Fahner** *
Department of Neuroinformatics
University of Bonn
Römerstr. 164
W-5300 Bonn 1, Germany

**Rolf Eckmiller**
Department of Neuroinformatics
University of Bonn
Römerstr. 164
W-5300 Bonn 1, Germany

## Abstract

Within a simple test-bed, application of feed-forward neurocontrol for short-term planning of robot trajectories in a dynamic environment is studied. The action network is embedded in a sensory-motoric system architecture that contains a separate world model. It is continuously fed with short-term predicted spatio-temporal obstacle trajectories, and receives robot state feedback. The action net allows for external switching between alternative planning tasks. It generates goal-directed motor actions – subject to the robot's kinematic and dynamic constraints – such that collisions with moving obstacles are avoided. Using supervised learning, we distribute examples of the optimal planner mapping over a structure-level adapted parsimonious higher order network. The training database is generated by a Dynamic Programming algorithm. Extensive simulations reveal, that the local planner mapping is highly nonlinear, but can be effectively and sparsely represented by the chosen powerful net model. Excellent generalization occurs for unseen obstacle configurations. We also discuss the limitations of feed-forward neurocontrol for growing planning horizons.

*Tel.: (228)–550–364    FAX: (228)–550–425    e–mail: gerald@nero.uni-bonn.de

# 1  INTRODUCTION

*Global* planning of goal directed trajectories subject to cluttered spatio-temporal, state-dependent constraints – as in the kinodynamic path planning problem (Donald, 1989) considered here – is a difficult task, probably best suited for systems with embedded sequential behavior; theoretical insights indicate that the related problem of connectedness is of unbounded order (Minsky, 1969). However, considering practical situations, there is a lack of globally disposable constraints at planning time, due to partially unmodelled environments. The question then arises, to what extent feed-forward neurocontrol may be effective for *local* planning horizons.

In this paper, we put aside problems of credit assignment, and world model identification. We focus on the complexity of representing a local version of the generic kinodynamic path planning problem by a feed-forward net. We investigate the capacity of sparse distributed planner representations to *generalize from example plans*.

# 2  ENVIRONMENT AND ROBOT MODELS

## 2.1  ENVIRONMENT

The world around the robot is a two-dimensional scene, occupied by obstacles moving all in parallel to the $y$-axis, with randomly choosen discretized $x$-positions, and with a continuous velocity spectrum. The environment's state is given by a list reporting position $(x_i, y_i) \in (\mathcal{X}, \mathcal{Y})$, $\mathcal{X} \in \{0, ..., 8\}$, $\mathcal{Y} = [y^-, y^+]$, and velocity $(0, v_i)$ ; $v_i \in [v^-, v^+]$ of each obstacle $i$. The environment dynamics is given by

$$y_i(t+1) \ = \ y_i(t) + v_i \ . \tag{1}$$

Obstacles are inserted at random positions, and with random velocities, into some region distant from the robot's workspace. At each time step, the obstacle's positions are updated according to eqn.(1), so that they will cross the robot's workspace some time.

## 2.2  ROBOT

We consider a point-like robot of unit mass, which is confined to move within some interval along the $x$-axis. Its state is denoted by $(x_r, \dot{x}_r) \in (\mathcal{X}, \dot{\mathcal{X}})$ ; $\dot{\mathcal{X}} = \{-1, 0, 1\}$. At each time step, a motor command $u \in \ddot{\mathcal{X}} = \{-1, 0, 1\}$ is applied to the robot. The robot dynamics is given by

$$\begin{aligned} \dot{x}_r(t+1) &= \dot{x}_r(t) + u(t) \\ x_r(t+1) &= x_r(t) + \dot{x}_r(t+1) \ . \end{aligned} \tag{2}$$

Notice that the set of admissible motor commands depends on the present robot state. With these settings, the robot faces a fluctuating number of obstacles crossing its baseline, similar to the situation of a pedestrian who wants to cross a busy street (Figure 1).

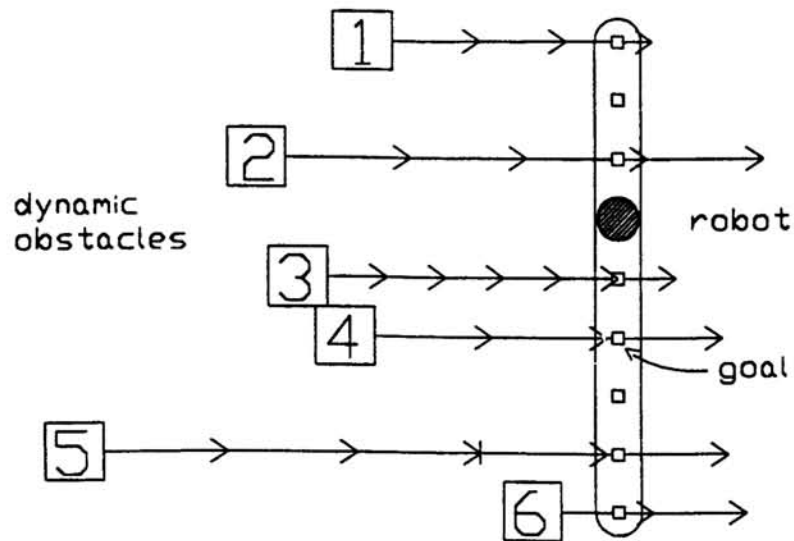

Figure 1: Obstacles Crossing the Robot's Workspace

# 3   SYSTEM ARCHITECTURE AND FUNCTIONALITY

Adequate modeling of the perception-action cycle is of decisive importance for the design of intelligent reactive systems. We partition the overall system into two modules: an active Perception Module (PM) with built-in capabilities for short-term environment forecasts, and a subsequent Action Module (AM) for motor command generation (Figure 2). Either module may be represented by a 'classical' algorithm, or by a neural net. PM is fed with a sensory data stream reporting the observed

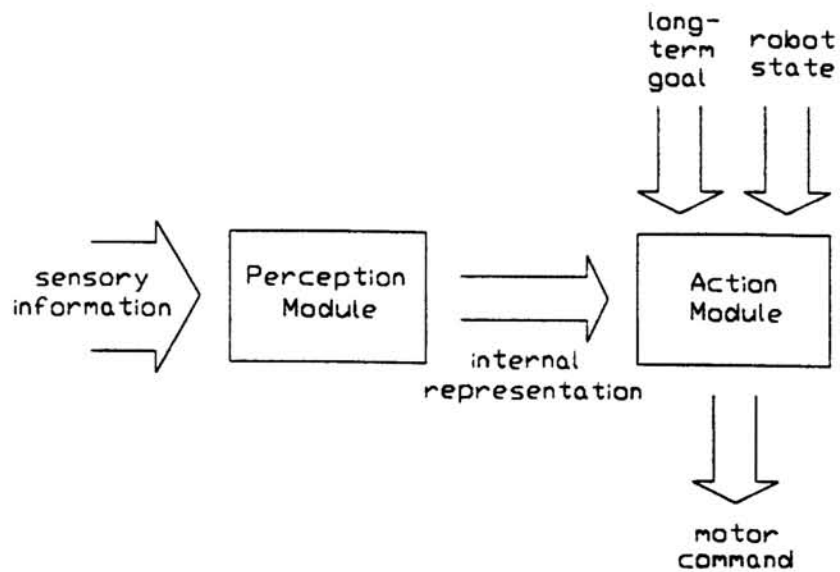

Figure 2: Sensory-Motoric System Architecture

dynamic scene of time-varying obstacle positions. From this, it assembles a spatio-

temporal internal representation of near-future obstacle trajectories. At each time step $t$, it actualizes the incidence function

$$occupancy(x, k) = \left\{ \begin{array}{lll} 1 & : & (x = x_i \ and \ -s < y_i(t+k) < s) \ \ for \ any \ obstacle \ i \\ -1 & : & otherwise, \end{array} \right.$$

where $s$ is some safety margin accounting for the $y$-extension of obstacles. The incidence function is defined on a spatio-temporal cone-shaped cell array, based at the actual robot position:

$$|x - x_r(t)| \le k \ ; \ k = 1, ..., HORIZON \tag{3}$$

The opening angle of this cone-shaped region is given by the robot's speed limit (here: one cell per time step). Only those cells that can potentially be reached by the robot within the local prediction-/planning horizon are thus represented by PM (see Figure 3). The functionality of AM is to map the current PM representation to

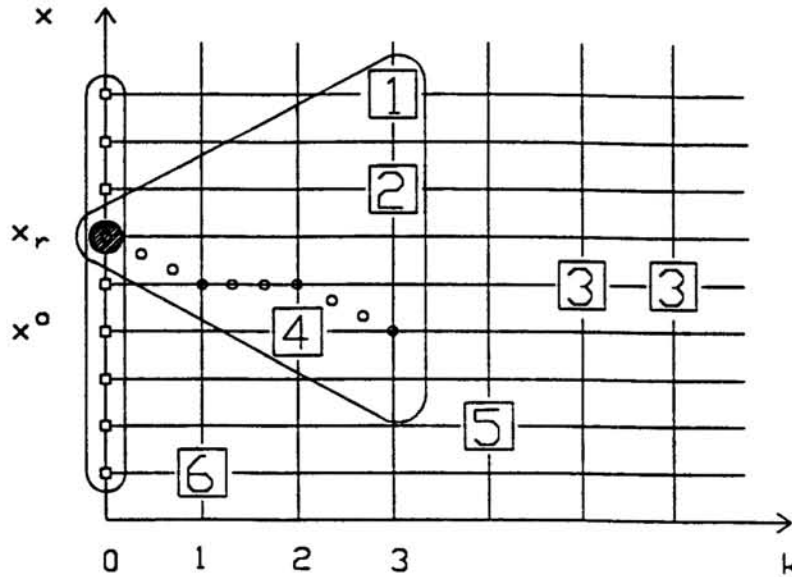

Figure 3: Space-Time Representation with Solution Path Indicated

an appropriate robot motor command, taking into account the present robot state, and paying regard to the currently specified long-term goal. Firstly, we realize the optimal AM by the Dynamic Programming (DP) algorithm (Bellman, 1957). Secondly, we use supervised learning to distribute optimal planning examples over a neural network.

## 4   DYNAMIC PROGRAMMING SOLUTION

Given PM's internal representation at time $t$, the present robot state, and some specification of the desired long-term goal, DP determines a sequence of motor commands minimizing some cost functional. Here we use

$$cost_{\{u(t),...,u(t+HORIZON)\}} = \sum_{k=0}^{HORIZON} (x_r(t+k) - x^o)^2 + c \, u(t+k)^2 , \tag{4}$$

with $x_r(t+k)$ given by the dynamics eqns.(2) (see solution path in Figure 3). By $x^o$, we denote the desired robot position or long-term goal. Deviations from this position are punished by higher costs, just as are costly accelerations. Obstacle collisions are excluded by restricting search to admissible cells $(x, \dot{x}, t+k)_{admissible}$ in phase-space-time (obeying $occupancy(x, t+k) = -1$). Training targets for time $t$ are constituted by the optimal present motor actions $u^{opt}(t)$, for which the minimum is attained in eqn.(4). For cases with degenerated optimal solutions, we consistently break symmetry, in order to obtain a deterministic target mapping.

## 5   NEURAL ACTION MODEL

For neural motor command generation, we use a single layer of structure-adapted *parsimonious Higher Order Neurons* (*parsi*HONs) (Fahner, 1992a, b), computing outputs $y_i \in [0, 1]$ ; $i = 1, 2, 3$. Target values for each single neuron are given by $y_i^{des} = 1$, if motor-action $i$ is the optimal one, otherwise, $y_i^{des} = 0$. As input, each neuron receives a bit-vector $\mathbf{x} = x_1, ..., x_N \in \{-1, 1\}^N$, whose components specify the values of PM's incidence function, the binary encoded robot state, and some task bits encoding the long-term goal. Using batch training, we maximize the log-likelihood criterion for each neuron independently. For recall, the motor command is obtained by a winner-takes-all decision: the index of the most active neuron yields the motor action applied.

Generally, atoms for nonlinear interactions within a bipolar-input HON are modelled by input monomials of the form

$$\eta_\alpha \equiv \prod_{j=1}^{N} x_j^{\alpha_j} \;\; ; \;\; \alpha = \alpha_1...\alpha_N \in \mathcal{R} \equiv \{0, 1\}^N . \tag{5}$$

Here, the $j^{th}$ bit of $\alpha$ is understood as exponent of $x_j$. It is well known that the complete set of monomials forms a basis for Boolean functions expansions (Karpovski, 1976). Combinatorial growth of the number of terms with increasing input dimension renders allocation of the complete basis impractical in our case. Moreover, an action model employing excessive numbers of basis functions would overfit trainig data, thus preventing generalization.

We therefore use a structural adaptation algorithm, as discussed in detail in (Fahner, 1992a, b), for automatic identification and inclusion of a sparse set of *relevant* nonlinearities present in the problem. In effect, this algorithm performs a guided stochastic search exploring the space of nonlinear interactions by means of an intertwined process of weight adaptation, and competition between nonlinear terms. The *parsi*HON model restricts the *number* of terms used, not their *orders*: instead of the exponential size set $\{\eta_\alpha : \alpha \in \mathcal{R}\}$, just a small subset $\{\eta_\beta : \beta \in \mathcal{S} \subset \mathcal{R}\}$ of terms is used within a parsimonious higher order function expansion

$$y^{est}(\mathbf{x}) = f\left[\sum_{\beta \in \mathcal{S}} w^\beta \eta_\beta(\mathbf{x})\right] \;\; ; \;\; w^\beta \in \mathbb{R} . \tag{6}$$

Here, $f$ denotes the usual sigmoid transfer function.

*parsi*HONs with high degrees of sparsity were effectively trained and emerged robust generalization for difficult nonlinear classification benchmarks (Fahner, 1992a, b).

# 6   SIMULATION RESULTS

We performed extensive simulations to evaluate the neural action network's capabilities to generalize from learned optimal planning examples. The planner was trained with respect to two alternative long-term goals: $x^o = 0$, or $x^o = 8$. Firstly, optimal DP planner actions were assembled over about $6,000$ time steps of the simulated environment (fairly crowded with moving obstacles), for both long-term goals. At each time step, optimal motor commands were computed for all $9 \times 3 = 27$ available robot states. From this bunch of situations we excluded those, where no collision-free path existed within the planning horizon considered: ($HORIZON = 3$). A total of 115,000 admissible training situations were left, out of the $6,000 \times 27 = 162,000$ one's generated . Thus, out of the full spectrum of robot states which were checked every time step, just about 19 states were not doomed to collide, at an average. These findings corrobate the difficulty of the choosen task.

Many repetitions are present in these accumulated patterns, reflecting the statistics of the simulated environment. We collapsed the original training set by removing repeated patterns, providing the learner with more information per pattern: a working data base containing about 20.000 different patterns was left.

Input to the neural action net consisted of a bit-vector of length $N = 21$, where $3 + 5 + 7$ bits encode PM's internal representation (cone size in Figure 3), 6 bits encode the robot's state, and a single task bit reports the desired goal. For training, we delimited single neuron learning to a maximum of 1000 epochs. In most cases, this was sufficient for successful training set classification for any of the three neurons ($y_i < .8$ for $y_i^{des} = 0$, and $y_i > .8$ for $y_i^{des} = 1$ ; $i = 1, 2, 3$). But even if some training patterns were misclassified by individual motor neurons, additional robustness stemming from the winner-takes-all decision rescued fault-free recall of the voting community. To test generalization of the neural action model, we par-

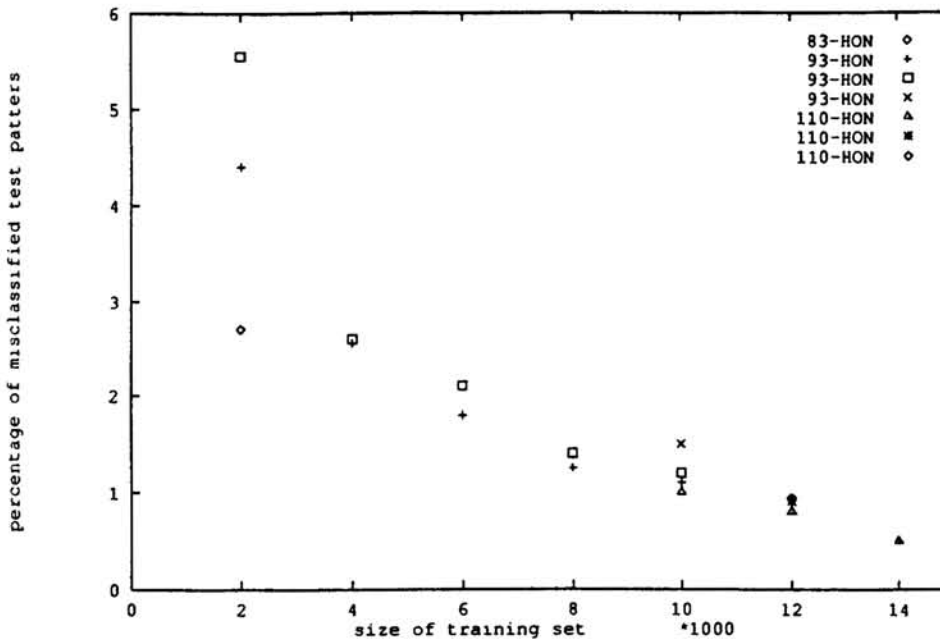

Figure 4: Generalization Behavior

titioned the data base into two parts, one containing training patterns, the other

containing new test patterns, not present in the training set. Several runs were performed with *parsi*HONs of sizes between 83 and 110 terms. Results for varying training set sizes are depicted in Figure 4. Test error decreases with increasing training set size, and falls as low as about one percent for about 12,000 training patterns. It continues to decrease for larger training sets. These findings corroborate that the trained architectures emerge sensible robust generalization.

To get some insight into the complexity of the mapping, we counted the number of terms which carry a given order. The resulting distribution has its maximum at order 3, exhibits many terms of orders 4 and higher, and finally decreases to zero for orders exceeding 10 (Figure 5). This indicates that the planner mapping considered is highly nonlinear.

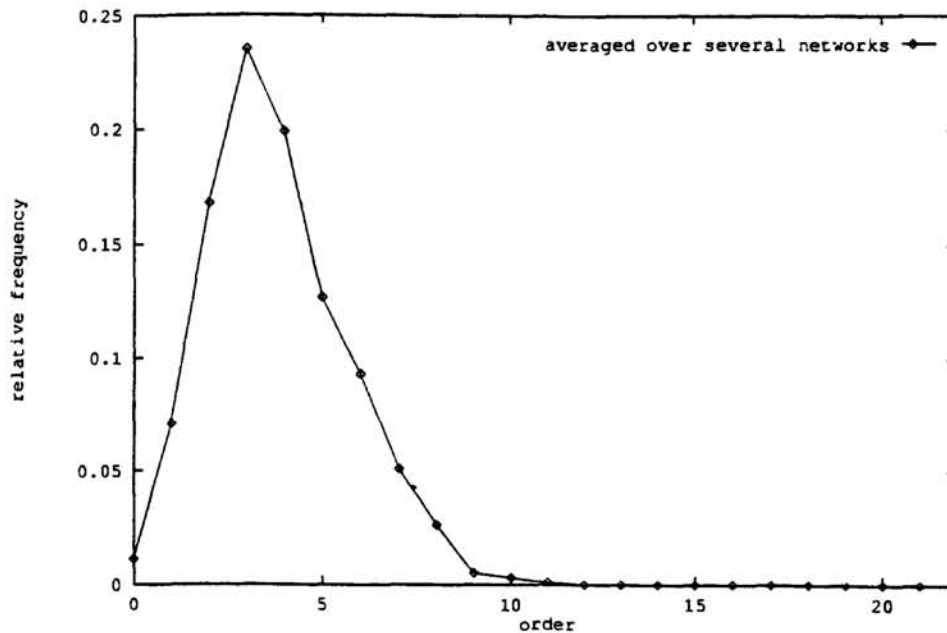

Figure 5: Distribution of Orders

## 7   DISCUSSION AND CONCLUSIONS

Sparse representation of planner mappings is desirable when representation of complete policy look-up tables becomes impracticable (Bellman's "curse of dimensionality"), or when computation of plans becomes expensive or conflicting with real-time requirements. For these reasons, it is urgent to investigate the capacity of neurocontrol for effective distributed representation and for robust generalization of planner mappings.

Here, we focused on a new type of shallow feed-forward action network for the local kinodynamic trajectory planning problem. An advantage with feed- forward nets is their low-latency recall, which is an important requirement for systems acting in rapidly changing environments. However, from theoretical considerations concerning the related problem of connectedness with its inherent serial character (Minsky, 1969), the planning problem under focus is expected to be hard for feed-forward nets. Even for rather local planning horizons, complex and nonlinear planner map-

pings must be expected. Using a powerful new neuron model that identifies the relevant nonlinearities inherent in the problem, we determined extremely parsimonious architectures for representation of the planner mapping. This indicates that some compact set of important features determines the optimal plan. The adapted networks emerged excellent generalization.

We encourage use of feed-forward nets for difficult local planning tasks, if care is taken that the models support effective representation of high-order nonlinearities. For growing planning horizons, it is expected that feed-forward neurocontrol will run into limitations (Werbos, 1992). The simple test-bed presented here would allow for insertion and testing also of other net models and system designs, including recurrent networks.

## Acknowledgements

This work was supported by Federal Ministry of Research and Technology (BMFT-project SENROB), grant 01 IN 105 A/0)

## References

E. B. Baum, F. Wilczek (1987). Supervised Learning of Probability Distributions by Neural Networks. In D. Anderson (Ed.), *Neural Information Processing Systems*, 52-61. Denver, CO: American Institute of Physics.

R. E. Bellman (1957). *Dynamic Programming*. Princeton University Press.

B. Donald (1989). *Near-Optimal Kinodynamic Planning for Robots With Coupled Dynamic Bounds*, Proc. IEEE Int. Conf. on Robotics and Automation.

G. Fahner, N. Goerke, R. Eckmiller (1992). *Structural Adaptation of Boolean Higher Order Neurons: Superior Classification with Parsimonious Topologies*, Proc. ICANN, Brighton, UK.

G. Fahner, R. Eckmiller. *Structural Adaptation of Parsimonious Higher Order Classifiers*, subm. to Neural Networks.

M. G. Karpovski (1976). *Finite Orthogonal Series in the Design of Digital Devices*. New York: John Wiley & Sons.

M. Minsky, S. A. Papert (1969). *Perceptrons*. Cambridge: The MIT Press.

P. Werbos (1992). Approximate Dynamic Programming for Real-Time Control and Neural Modeling. In D. White, D. Sofge (eds.) *Handbook of Intelligent Control*, 493-525. New York: Van Nostrand.